# Human Face Detection in Visual Scenes

**Henry A. Rowley**
har@cs.cmu.edu

**Shumeet Baluja**
baluja@cs.cmu.edu

**Takeo Kanade**
tk@cs.cmu.edu

School of Computer Science, Carnegie Mellon University, Pittsburgh, PA 15213, USA

## Abstract

We present a neural network-based face detection system. A retinally connected neural network examines small windows of an image, and decides whether each window contains a face. The system arbitrates between multiple networks to improve performance over a single network. We use a bootstrap algorithm for training, which adds false detections into the training set as training progresses. This eliminates the difficult task of manually selecting non-face training examples, which must be chosen to span the entire space of non-face images. Comparisons with another state-of-the-art face detection system are presented; our system has better performance in terms of detection and false-positive rates.

## 1 INTRODUCTION

In this paper, we present a neural network-based algorithm to detect frontal views of faces in gray-scale images. The algorithms and training methods are general, and can be applied to other views of faces, as well as to similar object and pattern recognition problems.

Training a neural network for the face detection task is challenging because of the difficulty in characterizing prototypical "non-face" images. Unlike in face *recognition*, where the classes to be discriminated are different faces, in face *detection*, the two classes to be discriminated are "images containing faces" and "images not containing faces". It is easy to get a representative sample of images which contain faces, but much harder to get a *representative sample* of those which do not. The size of the training set for the second class can grow very quickly.

We avoid the problem of using a huge training set of non-faces by selectively adding images to the training set as training progresses [Sung and Poggio, 1994]. This "bootstrapping" method reduces the size of the training set needed. Detailed descriptions of this training method, along with the network architecture are given in Section 2. In Section 3 the performance of the system is examined. We find that the system is able to detect 92.9% of faces with an acceptable number of false positives. Section 4 compares this system with a similar system. Conclusions and directions for future research are presented in Section 5.

## 2 DESCRIPTION OF THE SYSTEM

Our system consists of two major parts: a set of neural network-based filters, and a system to combine the filter outputs. Below, we describe the design and training of the filters,

which scan the input image for faces. This is followed by descriptions of algorithms for arbitrating among multiple networks and for merging multiple overlapping detections.

## 2.1   STAGE ONE: A NEURAL NETWORK-BASED FILTER

The first component of our system is a filter that receives as input a small square region of the image, and generates an output ranging from 1 to -1, signifying the presence or absence of a face, respectively. To detect faces anywhere in the input, the filter must be applied at every location in the image. To allow detection of faces larger than the window size, the input image is repeatedly reduced in size (by subsampling), and the filter is applied at each size. The set of scaled input images is known as an "image pyramid", and is illustrated in Figure 1. The filter itself must have some invariance to position and scale. The amount of invariance in the filter determines the number of scales and positions at which the filter must be applied.

With these points in mind, we can give the filtering algorithm (see Figure 1). It consists of two main steps: a preprocessing step, followed by a forward pass through a neural network. The preprocessing consists of lighting correction, which equalizes the intensity values across the window, followed by histogram equalization, which expands the range of intensities in the window [Sung and Poggio, 1994]. The preprocessed window is used as the input to the neural network. The network has retinal connections to its input layer; the receptive fields of each hidden unit are shown in the figure. Although the figure shows a single hidden unit for each subregion of the input, these units can be replicated. Similar architectures are commonly used in speech and character recognition tasks [Waibel *et al.*, 1989, Le Cun *et al.*, 1989].

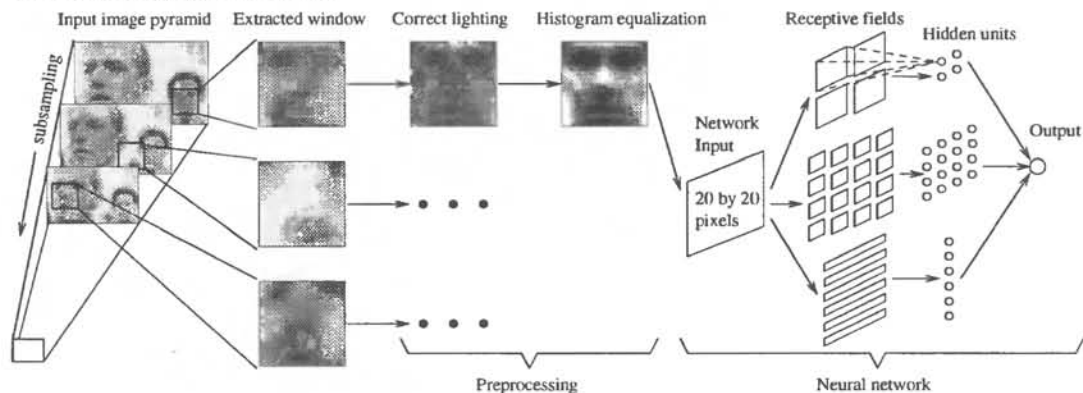

**Figure 1:** The basic algorithm used for face detection.

Examples of output from a single filter are shown in Figure 2. In the figure, each box represents the position and size of a window to which the neural network gave a positive response. The network has some invariance to position and scale, which results in multiple boxes around some faces. Note that there are some false detections; we present methods to eliminate them in Section 2.2. We next describe the training of the network which generated this output.

### 2.1.1   Training Stage One

To train a neural network to serve as an accurate filter, a large number of face and non-face images are needed. Nearly 1050 face examples were gathered from face databases at CMU and Harvard. The images contained faces of various sizes, orientations, positions, and intensities. The eyes and upper lip of each face were located manually, and these points were used to normalize each face to the same scale, orientation, and position. A 20-by-20 pixel region containing the face is extracted and preprocessed (by apply lighting correction and histogram equalization). In the training set, 15 faces were created from each original image, by slightly rotating (up to 10°), scaling (90%–110%), translating (up to half a pixel),

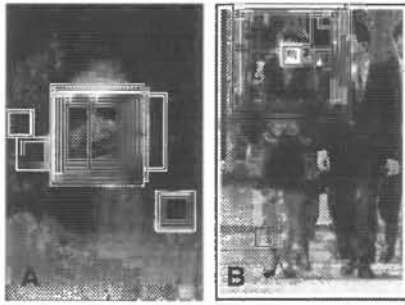

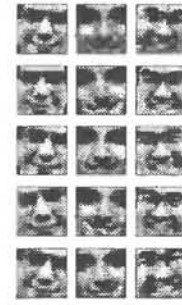

**Figure 2:** Images with all the above threshold detections indicated by boxes.

**Figure 3:** Example face images, randomly mirrored, rotated, translated, and scaled by small amounts.

and mirroring each face. A few example images are shown in Figure 3.

It is difficult to collect a *representative* set of non-faces. Instead of collecting the images before training is started, the images are collected during training, as follows [Sung and Poggio, 1994]:

1. Create 1000 non-face images using random pixel intensities.

2. Train a neural network to produce an output of 1 for the face examples, and -1 for the non-face examples.

3. Run the system on an image of scenery which contains no faces. Collect subimages in which the network incorrectly identifies a face (an output activation > 0).

4. Select up to 250 of these subimages at random, and add them into the training set. Go to step 2.

Some examples of non-faces that are collected during training are shown in Figure 4. We used 120 images for collecting negative examples in this bootstrapping manner. A typical training run selects approximately 8000 non-face images from the 146,212,178 subimages that are available at all locations and scales in the scenery images.

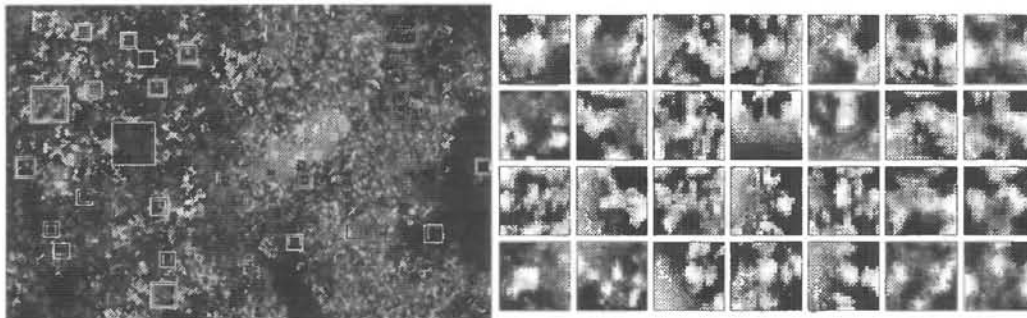

**Figure 4:** Some non-face examples which are collected during training.

## 2.2 STAGE TWO: ARBITRATION AND MERGING OVERLAPPING DETECTIONS

The examples in Figure 2 showed that just one network cannot eliminate all false detections. To reduce the number of false positives, we apply two networks, and use arbitration to produce the final decision. Each network is trained in a similar manner, with random initial weights, random initial non-face images, and random permutations of the order of presentation of the scenery images. The detection and false positive rates of the individual networks are quite close. However, because of different training conditions and because of self-selection of negative training examples, the networks will have different biases and will make different errors.

For the work presented here, we used very simple arbitration strategies. Each detection by a filter at a particular position and scale is recorded in an image pyramid. One way to

combine two such pyramids is by ANDing. This strategy signals a detection only if both networks detect a face at precisely the same scale and position. This ensures that, if a particular false detection is made by only one network, the combined output will not have that error. The disadvantage is that if an actual face is detected by only one network, it will be lost in the combination. Similar heuristics, such as ORing the outputs, were also tried.

Further heuristics (applied either before or after the arbitration step) can be used to improve the performance of the system. Note that in Figure 2, most faces are detected at multiple nearby positions or scales, while false detections often occur at single locations. At each location in an image pyramid representing detections, the number of detections within a specified neighborhood of that location can be counted. If the number is above a threshold, then that location is classified as a face. These detections are then collapsed down to a single point, located at their centroid. When this is done before arbitration, the centroid locations rather than the actual outputs from the networks are ANDed together.

If we further assume that a position is correctly identified as a face, then all other detections which overlap it are likely to be errors, and can therefore be eliminated. There are relatively few cases in which this heuristic fails; however, one such case is illustrated in the left two faces in Figure 2B, in which one face partially occludes another. Together, the steps of combining multiple detections and eliminating overlapping detections will be referred to as *merging detections*. In the next section, we show that by merging detections and arbitrating among multiple networks, we can reduce the false detection rate significantly.

## 3   EMPIRICAL RESULTS

A large number of experiments were performed to evaluate the system. Because of space restrictions only a few results are reported here; further results are presented in [Rowley *et al.*, 1995]. We first show an analysis of which features the neural network is using to detect faces, and then present the error rates of the system over two large test sets.

### 3.1   SENSITIVITY ANALYSIS

In order to determine which part of the input image the network uses to decide whether the input is a face, we performed a sensitivity analysis using the method of [Baluja and Pomerleau, 1995]. We collected a test set of face images (based on the training database, but with different randomized scales, translations, and rotations than were used for training), and used a set of negative examples collected during the training of an earlier version of the system. Each of the 20-by-20 pixel input images was divided into 100 two-by-two pixel subimages. For each subimage in turn, we went through the test set, replacing that subimage with random noise, and tested the neural network. The sum of squared errors made by the network is an indication of how important that portion of the image is for the detection task. Plots of the error rates for two networks we developed are shown in Figure 5.

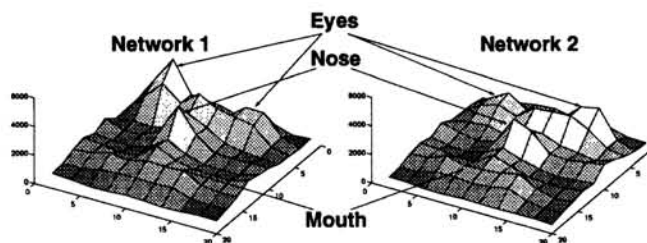

**Figure 5:** Sum of squared errors (z-axis) on a small test resulting from adding noise to various portions of the input image (horizontal plane), for two networks. Network 1 uses two sets of the hidden units illustrated in Figure 1, while network 2 uses three sets.

The networks rely most heavily on the eyes, then on the nose, and then on the mouth (Figure 5). Anecdotally, we have seen this behavior on several real test images: the network's accuracy decreases more when an eye is occluded than when the mouth is occluded. Further, when both eyes of a face are occluded, it is rarely detected.

## 3.2  TESTING

The system was tested on two large sets of images. Test Set A was collected at CMU, and consists of 42 scanned photographs, newspaper pictures, images collected from the World Wide Web, and digitized television pictures. Test set B consists of 23 images provided by Sung and Poggio; it was used in [Sung and Poggio, 1994] to measure the accuracy of their system. These test sets require the system to analyze 22,053,124 and 9,678,084 windows, respectively. Table 1 shows the performance for the two networks working alone, the effect of overlap elimination and collapsing multiple detections, and the results of using ANDing and ORing for arbitration. Each system has a better false positive rate (but a worse detection rate) on Test Set A than on Test Set B, because of differences in the types of images in the two sets. Note that for systems using arbitration, the ratio of false detections to windows examined is extremely low, ranging from 1 in 146,638 to 1 in 5,513,281, depending on the type of arbitration used. Figure 6 shows some example output images from the system, produced by merging the detections from networks 1 and 2, and ANDing the results. Using another neural network to arbitrate among the two networks gives about the same performance as the simpler schemes presented above [Rowley *et al.*, 1995].

**Table 1:** Detection and Error Rates

| Type | System | Test Set A | | Test Set B | |
|---|---|---|---|---|---|
| | | # miss / Detect rate | | # miss / Detect rate | |
| | | False detects / Rate | | False detects / Rate | |
| | 0) Ideal System | 0/169 | 100.0% | 0/155 | 100.0% |
| | | 0 | 0/22053124 | 0 | 0/9678084 |
| Single network, no heuristics | 1) Network 1 (52 hidden units, 2905 connections) | 17 | 89.9% | 11 | 92.9% |
| | | 507 | 1/43497 | 353 | 1/27417 |
| | 2) Network 2 (78 hidden units, 4357 connections) | 20 | 88.2% | 10 | 93.5% |
| | | 385 | 1/57281 | 347 | 1/27891 |
| Single network, with heuristics | 3) Network 1 → merge detections | 24 | 85.8% | 12 | 92.3% |
| | | 222 | 1/99338 | 126 | 1/76810 |
| | 4) Network 2 → merge detections | 27 | 84.0% | 13 | 91.6% |
| | | 179 | 1/123202 | 123 | 1/78684 |
| Arbitrating among two networks | 5) Networks 1 and 2 → AND → merge detections | 52 | 69.2% | 34 | 78.1% |
| | | 4 | 1/5513281 | 3 | 1/3226028 |
| | 6) Networks 1 and 2 → merge detections → AND | 36 | 78.7% | 20 | 87.1% |
| | | 15 | 1/1470208 | 15 | 1/645206 |
| | 7) Networks 1 and 2 → merge → OR → merge | 26 | 84.6% | 11 | 92.9% |
| | | 90 | 1/245035 | 64 | 1/151220 |

# 4  COMPARISON TO OTHER SYSTEMS

[Sung and Poggio, 1994] reports a face-detection system based on clustering techniques. Their system, like ours, passes a small window over all portions of the image, and determines whether a face exists in each window. Their system uses a supervised clustering method with six "face" and six "non-face" clusters. Two distance metrics measure the distance of an input image to the prototype clusters. The first metric measures the "partial" distance between the test pattern and the cluster's 75 most significant eigenvectors. The second distance metric is the Euclidean distance between the test pattern and its projection in the 75 dimensional subspace. These distance measures have close ties with Principal Components Analysis (PCA), as described in [Sung and Poggio, 1994]. The last step in their system is to use either a perceptron or a neural network with a hidden layer, trained to classify points using the two distances to each of the clusters (a total of 24 inputs). Their system is trained with 4000 positive examples, and nearly 47500 negative examples collected in the "bootstrap" manner. In comparison, our system uses approximately 16000 positive examples and 8000 negative examples.

Table 2 shows the accuracy of their system on Test Set B, along with the results of our

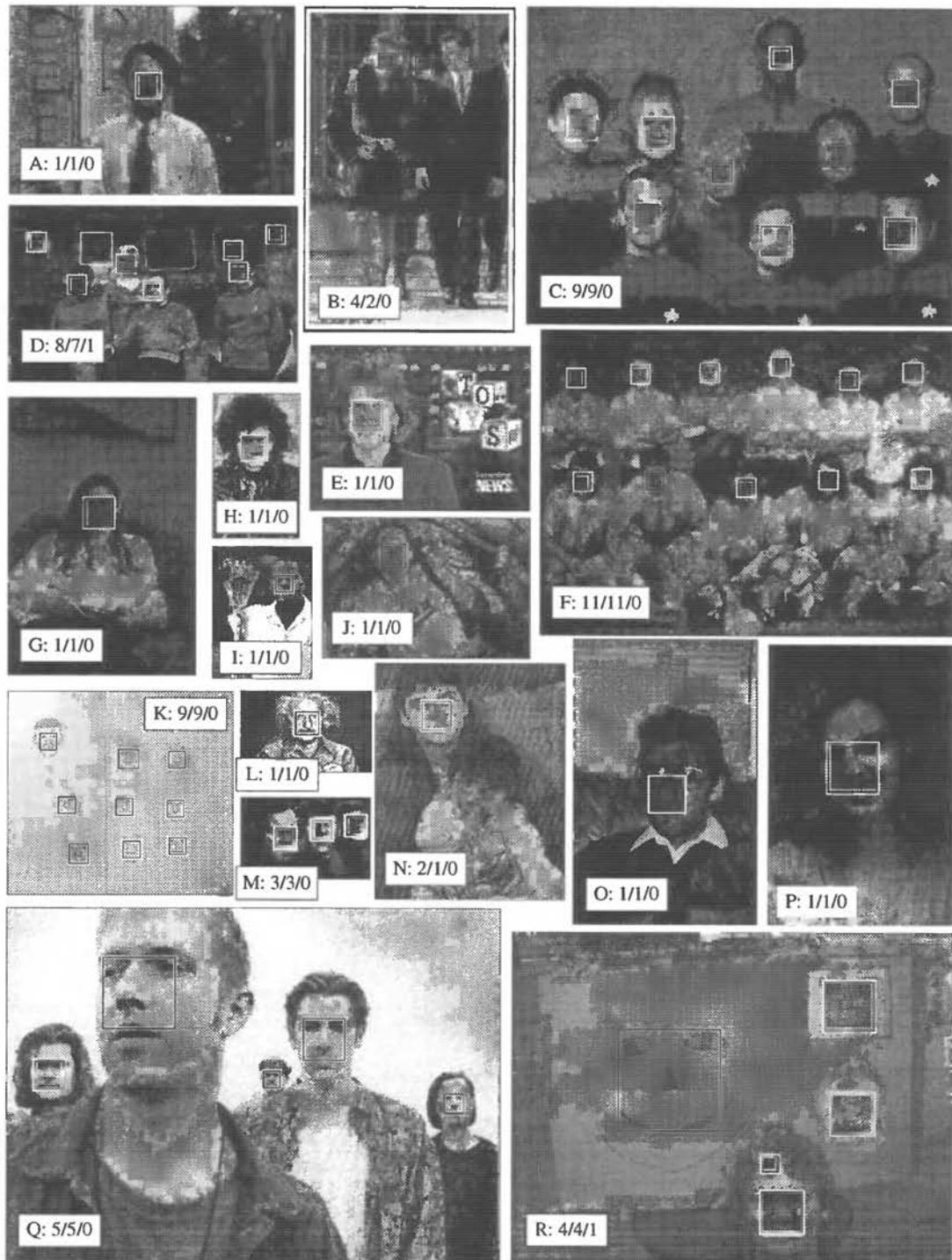

**Figure 6:** Output produced by System 6 in Table 1. For each image, three numbers are shown: the number of faces in the image, the number of faces detected correctly, and the number of false detections. Some notes on specific images: Although the system was not trained on hand-drawn faces, it detects them in K and R. One false detect is present in both D and R. Faces are missed in D (removed because a false detect overlapped it), B (one due to occlusion, and one due to large angle), and in N (babies with fingers in their mouths are not well represented in training data). Images B, D, F, K, L, and M were provided by Sung and Poggio at MIT. Images A, G, O, and P were scanned from photographs, image R was obtained with a CCD camera, images J and N were scanned from newspapers, images H, I, and Q were scanned from printed photographs, and image C was obtained off of the World Wide Web. Images P and B correspond to Figures 2A and 2B.

system using a variety of arbitration heuristics. In [Sung and Poggio, 1994], only 149 faces were labelled in the test set, while we labelled 155 (some are difficult for either system to detect). The number of missed faces is therefore six more than the values listed in their paper. Also note that [Sung and Poggio, 1994] check a slightly smaller number of windows over the entire test set; this is taken into account when computing the false detection rates. The table shows that we can achieve higher detection rates with fewer false detections.

**Table 2:** Comparison of [Sung and Poggio, 1994] and Our System on Test Set B

| System | Missed faces | Detect rate | False detects | Rate |
|---|---|---|---|---|
| 5) Networks 1 and 2 → AND → merge | 34 | 78.1% | 3 | 1/3226028 |
| 6) Networks 1 and 2 → merge → AND | 20 | 87.1% | 15 | 1/645206 |
| 7) Networks 1 and 2 → merge → OR → merge | 11 | 92.9% | 64 | 1/151220 |
| [Sung and Poggio, 1994] (Multi-layer network) | 36 | 76.8% | 5 | 1/1929655 |
| [Sung and Poggio, 1994] (Perceptron) | 28 | 81.9% | 13 | 1/742175 |

# 5  CONCLUSIONS AND FUTURE RESEARCH

Our algorithm can detect up to 92.9% of faces in a set of test images with an acceptable number of false positives. This is a higher detection rate than [Sung and Poggio, 1994]. The system can be made more conservative by varying the arbitration heuristics or thresholds.

Currently, the system does not use temporal coherence to focus attention on particular portions of the image. In motion sequences, the location of a face in one frame is a strong predictor of the location of a face in next frame. Standard tracking methods can be applied to focus the detector's attention. The system's accuracy might be improved with more positive examples for training, by using separate networks to recognize different head orientations, or by applying more sophisticated image preprocessing and normalization techniques.

### Acknowledgements

The authors thank Kah-Kay Sung and Dr. Tomaso Poggio (at MIT), Dr. Woodward Yang (at Harvard), and Michael Smith (at CMU) for providing training and testing images. We also thank Eugene Fink, Xue-Mei Wang, and Hao-Chi Wong for comments on drafts of this paper.

This work was partially supported by a grant from Siemens Corporate Research, Inc., and by the Department of the Army, Army Research Office under grant number DAAH04-94-G-0006. Shumeet Baluja was supported by a National Science Foundation Graduate Fellowship. The views and conclusions in this document are those of the authors, and should not be interpreted as necessarily representing official policies or endorsements, either expressed or implied, of the sponsoring agencies.

### References

[Baluja and Pomerleau, 1995] Shumeet Baluja and Dean Pomerleau. Encouraging distributed input reliance in spatially constrained artificial neural networks: Applications to visual scene analysis and control. Submitted, 1995.

[Le Cun *et al.*, 1989] Y. Le Cun, B. Boser, J. S. Denker, D. Henderson, R. E. Howard, W. Hubbard, and L. D. Jackel. Backpropagation applied to handwritten zip code recognition. *Neural Computation*, 1:541–551, 1989.

[Rowley *et al.*, 1995] Henry A. Rowley, Shumeet Baluja, and Takeo Kanade. Human face detection in visual scenes. CMU-CS-95-158R, Carnegie Mellon University, November 1995. Also available at http://www.cs.cmu.edu/~har/faces.html.

[Sung and Poggio, 1994] Kah-Kay Sung and Tomaso Poggio. Example-based learning for view-based human face detection. A.I. Memo 1521, CBCL Paper 112, MIT, December 1994.

[Waibel *et al.*, 1989] Alex Waibel, Toshiyuki Hanazawa, Geoffrey Hinton, Kiyohiro Shikano, and Kevin J. Lang. Phoneme recognition using time-delay neural networks. *Readings in Speech Recognition*, pages 393–404, 1989.